# The Effect of Singularities in a Learning Machine when the True Parameters Do Not Lie on Such Singularities

**Sumio Watanabe**

Precision and Intelligence Laboratory
Tokyo Institute of Technology
4259 Nagatsuta, Midori-ku, Yokohama, 226-8503 Japan
E-mail: *swatanab@pi.titech.ac.jp*

**Shun-ichi Amari**

Laboratory for Mathematical Neuroscience
RIKEN Brain Science Institute
Hirosawa, 2-1, Wako-shi, Saitama, 351-0198, Japan
E-mail: *amari@brain.riken.go.jp*

## Abstract

A lot of learning machines with hidden variables used in information science have singularities in their parameter spaces. At singularities, the Fisher information matrix becomes degenerate, resulting that the learning theory of regular statistical models does not hold. Recently, it was proven that, if the true parameter is contained in singularities, then the coefficient of the Bayes generalization error is equal to the pole of the zeta function of the Kullback information. In this paper, under the condition that the true parameter is almost but **not** contained in singularities, we show two results. (1) If the dimension of the parameter from inputs to hidden units is not larger than three, then there exits a region of true parameters where the generalization error is larger than those of regular models, however, if otherwise, then for any true parameter, the generalization error is smaller than those of regular models. (2) The symmetry of the generalization error and the training error does not hold in singular models in general.

## 1  Introduction

A lot of learning machines with hidden parts such as multi-layer perceptrons [8], gaussian mixtures[2], Boltzman machines, and Bayesian networks with latent variables [4] are nonidentifiable statistical models. In such learning machines, the mapping from the parameter to the probability distribution is not one-to-one. Moreover, they have complex singularities. In this paper, a parameter $w$ of a parametric probability density function $p(x|w)$ is called to be a singularity if and only if $\det I(w) = 0$,

where $I(w)$ is the Fisher information matrix at $w$. If a learning machine has singularities, then neither the maximum likelihood estimator nor the Bayes *a posteriori* distribution converges to the normal distribution in general [1][5].

Recently, despite of the mathematical difficulty of such learning machines, the asymptotic Bayes generalization error has been clarified using algebraic geometrical method [5][6]. The Bayes generalization error $G(n)$, which is defined as the average Kullback distance from the true distribution to the Bayes predictive distribution, is equal to

$$G(n) = \frac{\lambda}{n} + o(\frac{1}{n})$$

where $n$ is the number of training samples and $(-\lambda)$ is the rational number that is equal to the largest pole of the *zeta* function of the Kullback information and the prior [6][7]. If the true parameter is not a singular point, then $\lambda = d/2$, where $d$ is the dimension of the parameter space, whereas, if the set of the true parameters consists of singularities, then $\lambda$ is different from $d/2$ [6][8].

In almost learning machines, singularities of the parameter space correspond to smaller models contained in the parametric model. However, in practical applications, the true distribution is seldom contained completely in a finite model, and it often happens that the true parameter is almost but not completely contained in singularities.

In this paper, in order to clarify the effect of singularities when the true parameter lies in the neighborhood of singularities, we propose a new scaling method by which the Kullback distance from the singularities to the true distribution is equal to $c/n$, where $n$ is the number of training samples and $c$ is a controlling parameter. This scaling method, which is often used in comparing the powers of statistical hypothesis testing algorithms, enables us to clarify the effect of singularities.

We show two results. (1) If the number of the parameters from inputs to hidden units is not larger than three, then there exists $c > 0$ such that the generalization error is larger than those of the corresponding regular model. However, if otherwise, then for an arbitrary $c \geq 0$, the generalization error is made to be smaller by the singularities. (2) The symmetry of the generalization error and the training error does not hold in nonidentifiable learning machines in general.

## 2    A Singular Model

Since singularities in learning machines with hidden variables have quite complex geometrical structures in general, it needs the advanced method in modern algebraic geometry to treat them in a general manner [6]. In this paper, we study a simple hierarchical model. Even in this simple model, a universal phenomenon caused by singularities can be found. Let us consider a learning problem:

$$\text{Learner}: \quad p(y|\mathbf{x}, a, \mathbf{b}) \;=\; \frac{1}{\sqrt{2\pi}} \exp(-\frac{1}{2}(y - af(\mathbf{b}, \mathbf{x}))^2), \qquad (1)$$

$$\text{True}: \quad q(y|\mathbf{x}) \;=\; \frac{1}{\sqrt{2\pi}} \exp(-\frac{1}{2}(y - \frac{a_0}{\sqrt{n}}f(\mathbf{b}_0, \mathbf{x}))^2), \qquad (2)$$

where $y \in R^1$ is an output, $\mathbf{x} \in R^M$ is an input with the probability distribution $q(\mathbf{x})$. The parameter space is defined by $\{(a, \mathbf{b}) \in R^1 \times R^N\}$. The Kullback distance from $q(y|\mathbf{x})$ to $p(y|\mathbf{x}, a, \mathbf{b})$ is equal to $(1/2n) \, a_0^2 E_{\mathbf{x}}[f(\mathbf{b}_0, \mathbf{x})^2]$, where $E_{\mathbf{x}}$ denotes the expectation value over $\mathbf{x}$. If $f(0, \mathbf{x}) \equiv 0$, then an arbitrary point in $\{a = 0\} \cup \{\mathbf{b} = 0\}$ is a singularity. We assume that the *a priori* distribution $\varphi(a, \mathbf{b})$ is a $C^1$-class function and $\psi(\mathbf{b}) \equiv \varphi(0, \mathbf{b})$ has a compact support.

Let $D_n = \{(\mathbf{x}_i, y_i); i = 1, 2, \cdots, n\}$ be a set of training samples independently taken from $q(x)q(y|x)$. Both the Bayes *a posteriori* distribution $p(a, \mathbf{b}|D_n)$ and the Bayes predictive distribution $p(y|\mathbf{x}, D_n)$ are respectively defined by

$$p(a, \mathbf{b}|D_n) = \frac{1}{C_n} \varphi(a, \mathbf{b}) \prod_{i=1}^{n} p(y_i|\mathbf{x}_i, a, \mathbf{b}),$$

$$p(y|\mathbf{x}, D_n) = \int p(y|\mathbf{x}, a, \mathbf{b}) \, p(a, \mathbf{b}|D_n) \, da \, d\mathbf{b},$$

where $C_n$ is a normalizing constant. The generalization error $G(n)$ and the training error $T(n)$ are respectively defined by

$$\text{Generalization Error:} \quad G(n) = E\left[\log \frac{q(y_{n+1}|\mathbf{x}_{n+1})}{p(y_{n+1}|\mathbf{x}_{n+1}, D_n)}\right],$$

$$\text{Training Error:} \quad T(n) = E\left[\frac{1}{n} \sum_{k=1}^{n} \log \frac{q(y_k|\mathbf{x}_k)}{p(y_k|\mathbf{x}_k, D_n)}\right],$$

where $E$ shows the expectation value over all sets of training samples $D_n$ and the testing samples $(\mathbf{x}_{n+1}, y_{n+1})$. If the learning machine is a regular statistical model, then both $G(n) = d/(2n) + o(1/n)$ and $T(n) = -d/(2n) + o(1/n)$ hold, where $d$ is the dimension of the parameter space, hence the coefficient $d$ does not depend on the true parameter. In this paper, we show that this property does not hold in a singular learning machine.

We assume that the learning machine satisfies the condition

$$f(\mathbf{b}, \mathbf{x}) = \sum_{j=1}^{J} f_j(\mathbf{b}) e_j(\mathbf{x}) \tag{3}$$

where $\{e_j(\mathbf{x})\}$ is a set of orthonormal functions, $E_{\mathbf{x}}[e_i(\mathbf{x})e_j(\mathbf{x})] = \delta_{ij}$. Then it follows that $\|f(\mathbf{b})\|^2 \equiv \sum_{j=1} f_j(\mathbf{b})^2 = E_{\mathbf{x}}[f(\mathbf{b}, \mathbf{x})^2]$. Then we have the following theorem.

**Theorem 1** *The Bayes generalization and training errors can be asymptotically expanded as*

$$G(n) = \frac{\lambda(a_0, \mathbf{b}_0)}{2n} + o(\frac{1}{n}),$$

$$T(n) = \frac{\mu(a_0, \mathbf{b}_0)}{2n} + o(\frac{1}{n}).$$

*Here $\lambda(a_0, \mathbf{b}_0)$ and $\mu(a_0, \mathbf{b}_0)$ are constant functions of $n$ defined by*

$$\lambda(a_0, \mathbf{b}_0) = 1 + a_0^2\|f(\mathbf{b}_0)\|^2 - E_g\left[\sum_{j=1}^{J} a_0 f_j(\mathbf{b}_0)\frac{1}{Z(g)}\frac{\partial Z}{\partial g_j}\right]$$

$$\mu(a_0, \mathbf{b}_0) = \lambda(a_0, \mathbf{b}_0) - E_g\left[\sum_{j=1}^{J} 2g_j \frac{1}{Z(g)}\frac{\partial Z}{\partial g_j}\right]$$

*where $g = (g_j)$ is the $J$ dimensional gaussian distribution whose average and the covariance matrix are respectively zero and the identity, and $E_g$ shows the expectation value over $g$, and*

$$Z(g) = \int \exp\left[\frac{1}{2}\frac{1}{\|f(\mathbf{b})\|^2}\{\sum_{j=1}^{J}(g_j + a_0 f_j(\mathbf{b}_0))f_j(\mathbf{b})\}^2\right] \frac{\psi(\mathbf{b})}{\|f(\mathbf{b})\|} \, d\mathbf{b}.$$

**Proof of Theorem 1**. We use the rescaling parameter $\alpha = \sqrt{n}\, a$ and define the average $< S(\alpha, \mathbf{b}) >$ of a function of $S(\alpha, \mathbf{b})$ by

$$< S(\alpha, \mathbf{b}) >= \frac{\int \exp(-L(\alpha, \mathbf{b}))\; S(\alpha, \mathbf{b})\; \varphi(\alpha/\sqrt{n}, \mathbf{b})\; d\alpha\; d\mathbf{b}}{\int \exp(-L(\alpha, \mathbf{b}))\; \varphi(\alpha/\sqrt{n}, \mathbf{b})\; d\alpha\; d\mathbf{b}}$$

where, we use notations $d(\alpha, \mathbf{b}, \mathbf{x}) = \alpha f(\mathbf{b}, \mathbf{x}) - a_0 f(\mathbf{b}_0, \mathbf{x})$ and

$$L(\alpha, \mathbf{b}) = \frac{1}{n} \sum_{i=1}^{n} L_i(\alpha, \mathbf{b})$$

$$L_i(\alpha, \mathbf{b}) = \frac{1}{2}\, d(\alpha, \mathbf{b}, \mathbf{x}_i)^2 - \sqrt{n}\, \epsilon_i\, d(\alpha, \mathbf{b}, \mathbf{x}_i).$$

Here $\epsilon_i \equiv y_i - a_0 f(\mathbf{b}_0, x_i)/\sqrt{n}$ is a sample from the standard normal distribution. The Bayes generalization and training errors are respectively equal to

$$G(n) = E\left[ -\log < \exp\{ -\frac{L_{n+1}(\alpha, \mathbf{b})}{n} \} > \right]$$

$$T(n) = E\left[ -\frac{1}{n} \sum_{k=1}^{n} \log < \exp\{ -\frac{L_k(\alpha, \mathbf{b})}{n} \} > \right].$$

When $n \to \infty$, the central limiting theorem ensures the convergences in probability and in law respectively,

$$\frac{1}{n} \sum_{i=1}^{n} e_j(\mathbf{x}_i)\, e_k(\mathbf{x}_i) \to \delta_{jk}, \qquad \frac{1}{\sqrt{n}} \sum_{i=1}^{n} \epsilon_i\, e_j(\mathbf{x}_i) \to g_j,$$

where $g = (g_j)$ is subject to the normal distribution whose average and covariance matrix are respectively equal to zero and the identity. Then by using $\log(1 - t) = -t + t^2/2 + o(t^2)$ for small $t$, it follows that

$$\lim_{n \to \infty} 2nG(n) = \sum_{j=1}^{J} E_g\left[ \{ \frac{1}{Z} \frac{\partial Z}{\partial g_j} - a_0 f_j(\mathbf{b}_0) \}^2 \right],$$

$$\lim_{n \to \infty} 2nT(n) = \lim_{n \to \infty} 2nG(n) - 2E_g\left[ \sum_{j=1}^{J} g_j \frac{1}{Z} \frac{\partial Z}{\partial g_j} \right],$$

where $E_g$ shows the expectation value over the random variable $g$ and

$$Z(g) = \int \exp\left[ -\frac{1}{2} \sum_{j=1}^{J} \alpha^2 f_j(\mathbf{b})^2 + \sum_{j=1}^{J} \alpha f_j(\mathbf{b})(g_j + a_0 f_j(\mathbf{b}_0)) \right] \psi(\mathbf{b})\; d\alpha\; d\mathbf{b}.$$

By using the identity

$$\{ \frac{1}{Z} \frac{\partial Z}{\partial g_j} \}^2 = \frac{1}{Z} \frac{\partial^2 Z}{\partial g_j^2} - \frac{\partial}{\partial g_j} \{ \frac{1}{Z} \frac{\partial Z}{\partial g_j} \},$$

and $E_g[(\partial/\partial g_j)f(g)] = E_g[g_j f(g)]$ for an arbitrary function $f(g)$, we obtain Theorem 1. (**End of Proof**: Theorem 1).

Theorem 1 shows that, if $a_0 = 0$, then $\lambda(a_0, \mathbf{b}_0) = 1$, which coincides with the general theory for the case when the true parameter is contained in the singularities [6]. In fact, if $a_0 = 0$, the zeta function of the Kullback information

$$\zeta(z) = \int a^{2z} \|b\|^{2z}\, \varphi(a, b)\, da\, db,$$

has the largest pole at $z = -1/2$. The new point of this paper is that the learning coefficient $\lambda(a_0, \mathbf{b}_0)$ for $a_0 \neq 0, \mathbf{b}_0 \neq 0$ is obtained. Unfortunately it can not be represented by any simple function.

## 3 The Effect of Singularities

In order to study the effect of singularities, we adopt the simple learning machine,

$$af(\mathbf{b}, \mathbf{x}) = \sum_{j=1}^{N} ab_j e_j(\mathbf{x}) \tag{4}$$

where $a \in R^1$, $\mathbf{b} \in R^N$, $\mathbf{x} \in R^M$ ($N > 1$). Also we assume that $\psi(\mathbf{b})$ depends only the norm $\|\mathbf{b}\|$, that is to say, $\psi(\mathbf{b})$ can be rewritten as $\psi(\|\mathbf{b}\|)$. In this learning machine, if the true regression function is $y = 0$, then the set of true parameters is $\{(a, \mathbf{b}); a = 0 \text{ or } \mathbf{b} = 0\}$.

**Remark.** By using the re-parameterization $w_i = ab_i$, the learning machine eq.(4) results in

$$p(y|\mathbf{x}, \mathbf{w}) = \frac{1}{\sqrt{2\pi}} \exp(-\frac{1}{2}(y - \sum_{j=1}^{N} w_j e_j(\mathbf{x})))^2).$$

This learner is a regular statistical model, hence both $G(n) = N/(2n) + o(1/n)$ and $T(n) = -N/(2n) + o(1/n)$ hold. Therefore, by comparing $\lambda(a_0, \mathbf{b}_0)$ and $-\mu(a_0, \mathbf{b}_0)$ with $N$, let us clarify the effect of singularities.

**Theorem 2** *Let us consider the learning machine and the true distribution given by eq.(1) and eq.(2), which are restricted as eq.(4). If $N \geq 2$, then the Bayes generalization and training errors are respectively given by*

$$\lambda(a_0, \mathbf{b}_0) = 1 + E_g\left[(a_0^2\|\mathbf{b}_0\|^2 + a_0\mathbf{b}_0 \cdot g)\frac{Y_N(g)}{Y_{N-2}(g)}\right] \tag{5}$$

$$\mu(a_0, \mathbf{b}_0) = 1 - 2N + E_g\left[(a_0^2\|\mathbf{b}_0\|^2 + 3a_0\mathbf{b}_0 \cdot g + 2\|g\|^2)\frac{Y_N(g)}{Y_{N-2}(g)}\right] \tag{6}$$

*where*

$$Y_N(g) = \int_0^{\pi/2} d\theta \ \sin^N \theta \ \exp(-\frac{1}{2}\|a_0\mathbf{b}_0 + g\|^2 \sin^2 \theta).$$

**Proof of Theorem 2.** We introduce the general polar coordinate $\mathbf{b} = (r, \Omega)$. The function $Z(g)$ in Theorem 1 is given by

$$Z(g) = \int dr \int d\Omega \ \exp\{ \frac{((g + a_0\mathbf{b}_0) \cdot \Omega)^2}{2} \} \ \psi(r) \ r^{N-2}.$$

Therefore $Z(g)$ is independent of the direction of $g + a_0\mathbf{b}_0$, we can assume $g + a_0\mathbf{b}_0 = \|g + a_0\mathbf{b}_0\| \times (1, 0, \cdots, 0)$ without loss of generality. By representing $\Omega = \mathbf{b}/r$ as

$$b_i/r = \sin\theta_1 \cdots \sin\theta_{i-1} \cos\theta_i \quad (1 \leq i \leq N-1),$$
$$b_N/r = \sin\theta_1 \cdots \sin\theta_{N-1},$$

we obtain

$$Z(g) = const. \int_0^{\pi/2} \sin^{N-2}\theta_1 \exp(\frac{\|a_0\mathbf{b}_0 + g\|^2}{2} \cos^2\theta_1) \ d\theta_1.$$

which completes the proof. (**End of Proof**: Theorem 2).

Unfortunately, the function $\lambda(a_0, \mathbf{b}_0)$ in eq.(5) can not be represented by any classically analytic function. Figure 1 shows the value $\lambda(a_0, \mathbf{b}_0)$ given by eq.(5) by numerical calculations, for the cases $N = 2, 3, .., 6$. The horizontal and longitudinal lines respectively show $|a_0|\|\mathbf{b}_0\|$ and $\lambda(a_0, \mathbf{b}_0)/N$. The generalization error

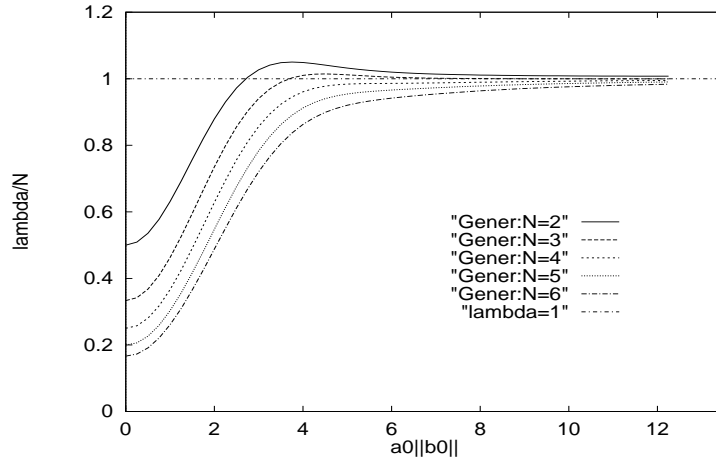

Figure 1: Coefficients of Generalization Errors $\lambda(a_0, b_0)/N$ for $a_0\|b_0\|$

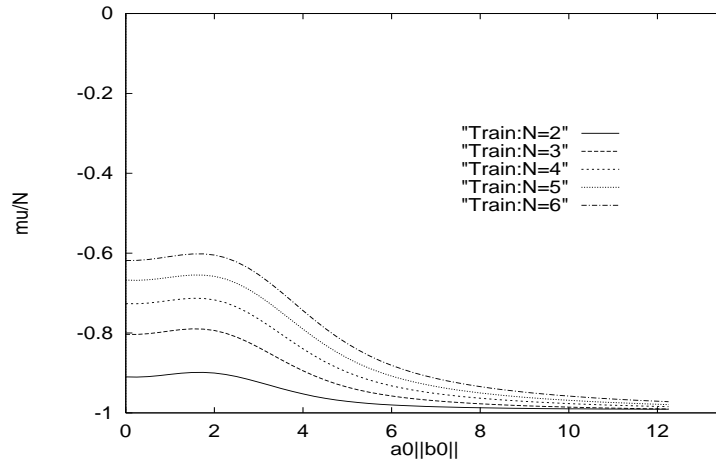

Figure 2: Coefficients of Training Errors $\mu(a_0, b_0)/N$ for $a_0\|b_0\|$

is smaller than that of the corresponding regular statistical model if and only if $\lambda(a_0, \mathbf{b}_0)/N < 1$.

For all cases $2 \leq N \leq 6$, $\lambda(a_0, \mathbf{b}_0)$ converges to the dimension $N$ when $|a_0|\|\mathbf{b}_0\| \to \infty$. If $N = 2$ and $N = 3$, $\lambda(a_0, \mathbf{b}_0)$ becomes larger than $N$, if the true parameter mismatches the singularities. When $N = 2$, in the region $|a_0|\|\mathbf{b}_0\| > 2.8$, $\lambda(a_0, \mathbf{b}_0) > N$. When $N = 3$, only in the interval $3.8 < |a_0|\|\mathbf{b}_0\| < 6.8$, $\lambda(a_0, \mathbf{b}_0) > N$.

On the other hand, if $N \geq 4$, the learning coefficient $\lambda(a_0, \mathbf{b}_0)$ is always smaller than $N$, even if the true parameter is not contained in singularities. If the dimension of the parameter is large, then singularities make the Bayes generalization error smaller than regular statistical models, independently of the place of the true parameter.

This result can be analyzed more precisely by the asymptotic expansion.

**Theorem 3** *The coefficients can be asymptotically expanded when* $|a_0|\|\mathbf{b}_0\| \to \infty$.

$$\lambda(a_0, \mathbf{b}_0) = N - \frac{(N-1)(N-3)}{a_0^2\|\mathbf{b}_0\|^2} + o(\frac{1}{a_0^2\|\mathbf{b}_0\|^2}),$$

$$\mu(a_0, \mathbf{b}_0) = -N + \frac{(N-1)^2}{a_0^2\|\mathbf{b}_0\|^2} + o(\frac{1}{a_0^2\|\mathbf{b}_0\|^2}).$$

In this theorem, $a_0^2\|b_0\|^2/2$ is equal to the Kullback distance from the singularities to the true distribution. It should be emphasized that the symmetrical relation $\lambda(a_0, \mathbf{b}_0) + \mu(a_0, \mathbf{b}_0) = 0$ does not hold near the singularities. In the generalization error, the coefficient of $1/a_0^2\|\mathbf{b}_0\|^2$ is positive if $N = 2$, whereas it is negative if $N \geq 4$. When $N = 3$, then the coefficient is equal to zero.

**Proof of Theorem 3** The function $Y_N(g)$ in Theorem 2 is rewritten as

$$Y_N(g) = \frac{1}{\|a_0\mathbf{b}_0 + g\|^2} \int_0^1 \frac{x^N}{\sqrt{1 - \frac{x^2}{\|a_0\mathbf{b}_0 + g\|^2}}} \exp(-\frac{x^2}{2})dx$$

Then by using

$$\frac{1}{\sqrt{1 - \frac{x^2}{\|a_0\mathbf{b}_0 + g\|^2}}} \cong 1 + \frac{x^2}{2\|a_0\mathbf{b}_0 + g\|^2},$$

we have an asymptotic expansion,

$$\lambda(a_0, \mathbf{b}_0) = 1 + E_g\left[ (a_0^2\|\mathbf{b}_0\|^2 + a_0\mathbf{b}_0 \cdot g) \frac{\frac{C_N}{\|a_0\mathbf{b}_0 + g\|^{M+1}} + \frac{C_{N+2}}{2\|a_0\mathbf{b}_0 + g\|^{M+3}}}{\frac{C_{N-2}}{\|a_0\mathbf{b}_0 + g\|^{M-1}} + \frac{C_N}{2\|a_0\mathbf{b}_0 + g\|^{M+1}}} \right],$$

where $C_N = 2^{(N-1)/2}\Gamma(\frac{N+1}{2})$. The training error can be obtained by the same way.
(**End of Proof**: Theorem 3).

## 4 Discussion

Let us shortly discuss three points.

Firstly, in this paper, we compared a simple layered model with a regular statistical model. If we employ a linear learner

$$y = \sum_{j=1}^{N} b_j e_j(\mathbf{x}),$$

then we can expect the more precise statistical estimation by making it to be the hierarchical model,

$$y = \sum_{j=1}^{N} ab_j e_j(\mathbf{x}),$$

if $N \geq 4$ and Bayesian estimation is applied.

Secondly, the Bayesian model selection is usually carried out by minimizing the stochastic complexities,

$$F(D_n) = -\log \int \prod_{i=1}^{n} p(y_i|\mathbf{x}_i, a, \mathbf{b})\varphi(a, \mathbf{b})\, da\mathbf{b}.$$

Let us consider the model selection problem, the model $y = 0$ or the model in eq.(1). If the Kullback distance from the singularities to the true paramater is equal to $c/n$ and if $n$ is sufficiently large, then for an arbitrary $c$, $y = 0$ is selected with the probability one. Theoretically speaking, this fact shows that the minimum stochastic complexity criterion is not equivalent to the minimum generalization error criterion.

And lastly, we have shown that, if the true parameter is at the neighborhood of singularities, then the symmetry of the generalization error and the training error does not hold. Therefore the generalization error can not be estimated based on the training error using the conventional method.

These three points are the important problems for future study.

## 5    Conclusion

Effect of singularities when the true parameter mismatches them is clarified. Singularities make the Bayes generalization error to be small if the dimension of the inputs to hidden units is large. We expect that this research will be a base to clarify the reason why neural information processing systems need hierarchical structures.

This work was supported by the Ministry of Education, Science, Sports, and Culture in Japan, Grant-in-aid for scientific research 12680370.

## References

[1] Amari,S., Park,H., and Ozeki,T. (2002) Geometrical singularities in the neuro-manifold of multilayer perceptrons. *Advances in Neural Information Processing Systems*, Vol.14.

[2] Hartigan, J.A. (1985) A Failure of likelihood asymptotics for normal mixtures. *Proceedings of the Berkeley Conference in Honor of J.Neyman and J.Kiefer*, Vol.2, pp.807-810.

[3] Hironaka, H. (1964). Resolution of singularities of an algebraic variety over a field of characteristic zero. *Annals of Mathematics*, 79, 109-326.

[4] Rusakov, D, Geiger,D.(2002) Asymptotic model selection for naive Bayesian networks. *Proc. of UAI02*.

[5] Watanabe, S. (1999). Algebraic analysis for singular statistical estimation. *Lecture Notes in Computer Science*, 1720, 39-50.

[6] Watanabe, S.,(2001) Algebraic analysis for nonidentifiable learning machines. *Neural Computation*, 13,(4), pp.899-933.

[7] Watanabe, S. (2001) Algebraic information geometry for learning machines with singularities. *Advances in Neural Information Processing Systems*, Vol.13, 329-336.

[8] Watanabe, S. (2001) Algebraic geometrical methods for hierarchical learning machines. International Journal of Neural Networks, Vol.14, No.8, 1049-1060.

[9] Watanabe,S., & Amari,S.-I.(2003) Learning coefficients of layered models when the true distriburion mismatches the singularities.*Neural Computation*, to appear.
